# Probabilistic Inference of Hand Motion from Neural Activity in Motor Cortex

**Y. Gao**[*]   **M. J. Black**[†]   **E. Bienenstock**[*§]   **S. Shoham**[¶]   **J. P. Donoghue**[§]

[*]Division of Applied Mathematics, Brown University, Providence, RI 02912
[†]Dept. of Computer Science, Brown University, Box 1910, Providence, RI 02912
[¶]Princeton University, Dept. of Molecular Biology Princeton, NJ, 08544
[§]Dept. of Neuroscience, Brown University, Providence, RI 02912

*gao@cfm.brown.edu*, *black@cs.brown.edu*, *elie@dam.brown.edu*,
*sshoham@princeton.com*, *john_donoghue@brown.edu*

## Abstract

*Statistical learning and probabilistic inference techniques are used to infer the hand position of a subject from multi-electrode recordings of neural activity in motor cortex. First, an array of electrodes provides training data of neural firing conditioned on hand kinematics. We learn a non-parametric representation of this firing activity using a Bayesian model and rigorously compare it with previous models using cross-validation. Second, we infer a posterior probability distribution over hand motion conditioned on a sequence of neural test data using Bayesian inference. The learned firing models of multiple cells are used to define a non-Gaussian likelihood term which is combined with a prior probability for the kinematics. A particle filtering method is used to represent, update, and propagate the posterior distribution over time. The approach is compared with traditional linear filtering methods; the results suggest that it may be appropriate for neural prosthetic applications.*

## 1  Introduction

This paper explores the use of statistical learning methods and probabilistic inference techniques for modeling the relationship between the motion of a monkey's arm and neural activity in motor cortex. Our goals are threefold: (i) to investigate the nature of encoding in motor cortex, (ii) to characterize the probabilistic relationship between arm kinematics (hand position or velocity) and activity of a simultaneously recorded neural population, and (iii) to optimally reconstruct (decode) hand trajectory from population activity to smoothly control a prosthetic robot arm (cf [14]).

A multi-electrode array (Figure 1) is used to simultaneously record the activity of 24 neurons in the arm area of primary motor cortex (MI) in awake, behaving, macaque monkeys. This activity is recorded while the monkeys manually track a smoothly and randomly mov-

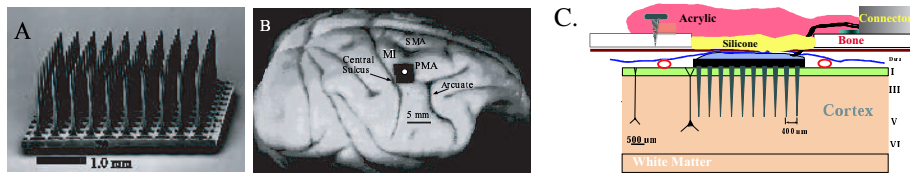

Figure 1: Multi-electrode array. *A*. 10X10 matrix of electrodes. Separation $400\mu$m (size 4X4 mm). *B*. Location of array in the MI arm area. *C*. Illustration of implanted array (courtesy N. Hatsopoulos).

ing visual target on a computer monitor [12]. Statistical learning methods are used to derive Bayesian estimates of the conditional probability of firing for each cell given the kinematic variables (we consider only hand velocity here). Specifically, we use non-parametric models of the conditional firing, learned using regularization (smoothing) techniques with cross-validation. Our results suggest that the cells encode information about the position and velocity of the hand in space. Moreover, the non-parametric models provide a better explanation of the data than previous parametric models [6, 10] and provide new insight into neural coding in MI.

Decoding involves the inference of the hand motion from the firing rate of the cells. In particular, we represent the posterior probability of the entire hand trajectory conditioned on the observed sequence of neural activity (spike trains). The nature of this activity results in ambiguities and a non-Gaussian posterior probability distribution. Consequently, we represent the posterior non-parametrically using a discrete set of samples [8]. This distribution is predicted and updated in non-overlapping 50 ms time intervals using a Bayesian estimation method called particle filtering [8]. Experiments with real and synthetic data suggest that this approach provides probabilistically sound estimates of kinematics and allows the probabilistic combination of information from multiple neurons, the use of priors, and the rigorous evaluation of models and results.

## 2 Methods: Neural Recording

The design of the experiment and data collection is described in detail in [12]. Summarizing, a ten-by-ten array of electrodes is implanted in the primary motor cortex (MI) of a Macaque monkey (Figure 1) [7, 9, 12]. Neural activity in motor cortex has been shown to be related to the movement kinematics of the animal's arm and, in particular, to the direction of hand motion [3, 6]. Previous behavioral tasks have involved reaching in one of a fixed number of directions [3, 6, 14]. To model the relationship between continuous, smooth, hand motion and neural activity, we use a more complex scenario where the monkey performs a continuous tracking task in which the hand is moved on a 2D tablet while holding a low-friction manipulandum that controls the motion of a feedback dot viewed on a computer monitor (Figure 2*a*) [12]. The monkey receives a reward upon completion of a successful trial in which the manipulandum is moved to keep the feedback dot within a pre-specified distance of the target. The path of the target is chosen to be a smooth random walk that effectively samples the space of hand positions and velocities: measured hand positions and velocities have a roughly Gaussian distribution (Figure 2*b* and *c*) [12]. Neural activity is amplified, waveforms are thresholded, and spike sorting is performed off-line to isolate the activity of individual cells [9]. Recordings from 24 motor cortical cells are measured simultaneously with hand kinematics.

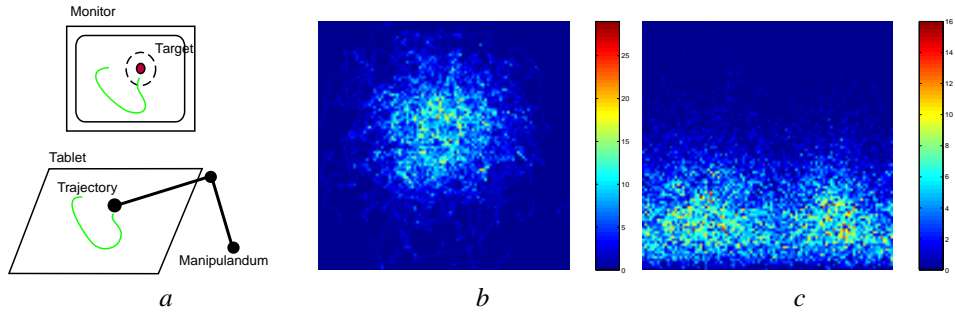

Figure 2: Smooth tracking task. (*a*) The target moves with a smooth random walk. Distribution of the position (*b*) and velocity (*c*) of the hand. Color coding indicates the frequency with which different parts of the space are visited. (*b*) Position: horizontal and vertical axes represent $x$ and $y$ position of the hand. (*c*) Velocity: the horizontal axis represents direction, $-\pi \leq \theta < \pi$, and the vertical axis represents speed, $r$.

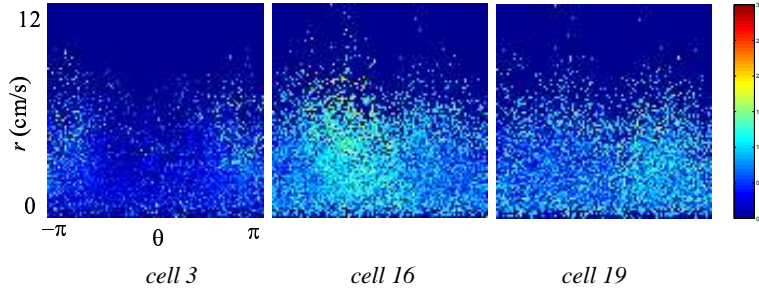

*cell 3*         *cell 16*         *cell 19*

Figure 3: Observed mean conditional firing rates in 50 ms intervals for three cells given hand velocity. The horizontal axis represents the direction of movement, $\theta$, in radians ("wrapping" around from $-\pi$ to $\pi$). The vertical axis represents speed, $r$, and ranges from 0 cm/s to 12 cm/s. Color ranges from dark blue (no measurement) to red (approximately 3 spikes).

## 3   Modeling Neural Activity

Figure 3 shows the measured mean firing rate within 50 ms time intervals for three cells conditioned on the subject's hand velocity. We view the neural firing activity in Figure 3 as a stochastic and sparse realization of some underlying model that relates neural firing to hand motion. Similar plots are obtained as a function of hand position. Each plot can be thought of as a type of "tuning function" [12] that characterizes the response of the cell conditioned on hand velocity. In previous work, authors have considered a variety of models of this data including a cosine tuning function [6] and a modified cosine function [10]. Here we explore a non-parametric model of the underling activity and, adopting a Bayesian formulation, seek a maximum *a posterior* (MAP) estimate of a cell's conditional firing.

Adopting a Markov Random Field (MRF) assumption [4], let the velocity space, $\mathbf{v} = [r, \theta]^T$, be discretized on a $100 \times 100$ grid. Let $\mathbf{g}$ be the array of true (unobserved) conditional neural firing and $\mathbf{f}$ be the corresponding observed mean firing. We seek the posterior probability

$$p(\mathbf{g} \mid \mathbf{f}) = \Pi_{\mathbf{v}} \left( \kappa \, p(f_{\mathbf{v}} \mid g_{\mathbf{v}}) \, \Pi_{i=1}^{\eta} p(g_{\mathbf{v}} \mid g_{\mathbf{v}_i}) \right) \qquad (1)$$

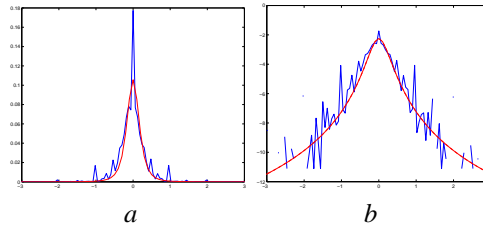

Figure 4: Prior probability of firing variation ($\Delta g$). *(a)* Probability of firing variation computed from training data (blue). Proposed robust prior model (red) plotted for $\sigma = 0.28$. *(b)* Logarithm of the distributions shown to provide detail.

where $\kappa$ is a normalizing constant independent of $\mathbf{g}$, $f_{\mathbf{v}}$ and $g_{\mathbf{v}}$ are the observed and true mean firing at velocity $\mathbf{v}$ respectively, $g_{\mathbf{v}_i}$ represents the firing rate for the $i^{\text{th}}$ neighboring velocity of $\mathbf{v}$, and the neighbors are taken to be the four nearest velocities ($\eta = 4$).

The first term on the right hand side represents the *likelihood* of observing a particular firing rate $f_{\mathbf{v}}$ given that the true rate is $g_{\mathbf{v}}$. Here we compare two *generative models* of the neural spiking process within 50 ms; a Poisson model, $p_P$, and a Gaussian model, $p_G$:

$$p_P(f \mid g) = \frac{1}{f!}\, g^f e^{-g}, \quad p_G(f \mid g) = \frac{1}{\sqrt{2\pi}\sigma} \exp\left( -\frac{(f-g)^2}{2\sigma^2} \right).$$

The second term is a spatial *prior* probability that encodes our expectations about $\Delta g$, the variation of neural activity in velocity space. The MRF prior states that the firing, $g_{\mathbf{v}}$, at velocity $\mathbf{v}$ depends only on the firing at neighboring velocities. We consider two possible prior models for the distribution of $\Delta g$: Gaussian and "robust". A Gaussian prior corresponds to an assumption that the firing rate varies smoothly. A robust prior assumes a heavy-tailed distribution of the spatial variation (see Figure 4), $\Delta g$, (derivatives of the firing rate in the $r$ and $\theta$ directions) and implies piecewise smooth data. The two spatial priors are

$$p_R(\Delta g) = \frac{2\sigma^3}{\pi(\sigma^2 + \Delta g^2)^2}, \quad p_G(\Delta g) = \frac{1}{\sqrt{2\pi}\sigma} \exp\left( -\frac{(\Delta g)^2}{2\sigma^2} \right).$$

The various models (cosine, a modified cosine (Moran and Schwartz [10]), Gaussian+Gaussian, and Poisson+Robust) are fit to the training data as shown in Figure 5.[1] In the case of the Gaussian+Gaussian and Poisson+Robust models, the optimal value of the $\sigma$ parameter is computed for each cell using cross validation. During cross-validation, each time 10 trials out of 180 are left out for testing and the models are fit with the remaining training data. We then compute the log likelihood of the test data given the model. This provides a measure of how well the model captures the statistical variation in the training set and is used for quantitative comparison. The whole procedure is repeated 18 times for different test data sets.

The solution to the Gaussian+Gaussian model can be computed in closed form but for the Poisson+Robust model no closed form solution for $\mathbf{g}$ exists and an optimal Bayesian estimate could be achieved with simulated annealing [4]. Instead, we derive an approximate

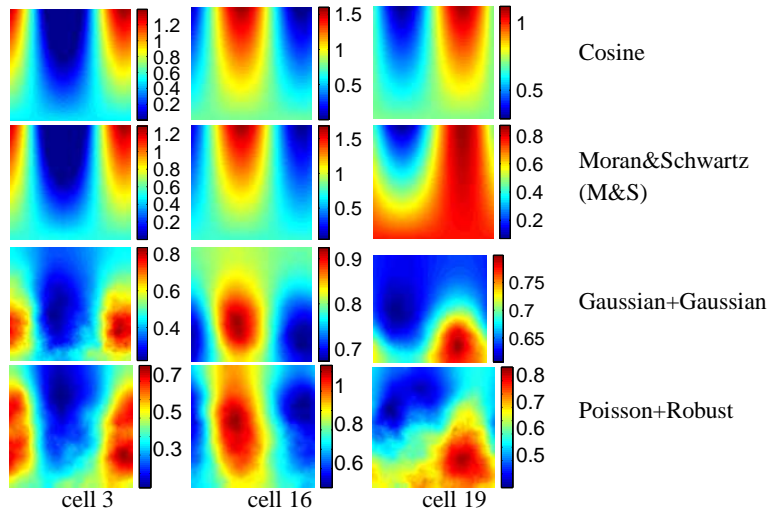

| | | | Cosine |
| | | | Moran&Schwartz (M&S) |
| | | | Gaussian+Gaussian |
| | | | Poisson+Robust |

cell 3      cell 16      cell 19

Figure 5: Estimated firing rate for cells in Figure 3 using different models.

| Method: | Log Likelihood Ratio | p-value |
|---|---|---|
| G+G over Cosine | 24.9181 | 7.6294e-06 |
| G+G over M&S | 15.8333 | 0.0047 |
| P+R over Cosine | 50.0685 | 7.6294e-06 |
| P+R over M&S | 32.2218 | 7.6294e-06 |

Table 1: Numerical comparison; log likelihood ratio of pairs of models and the significance level given by Wilcoxon signed rank test (Splus, MathSoft Inc., WA).

solution for $\mathbf{g}$ in (1) by minimizing the negative logarithm of the distribution using standard regularization techniques [1, 13] with missing data, the learned prior model, and a Poisson likelihood term [11]. Simple gradient descent [1] with deterministic annealing provides a reasonable solution. Note that the negative logarithm of the prior term can be approximated by the robust statistical error function $\rho(\Delta g) = \Delta g/(\sigma^2 + (\Delta g)^2)$ which has been used extensively in machine vision and image processing for smoothing data with discontinuities [1, 5].

Figure 5 shows the various estimates of the receptive fields. Observe that the pattern of firing is not Gaussian. Moreover, some cells appear to be tuned to motion direction, $\theta$, and not to speed, $r$, resulting in vertically elongated patterns of firing. Other cells (e.g. cell 19) appear to be tuned to particular directions and speeds; this type of activity is not well fit by the parametric models.

Table 1 shows a quantitative comparison using cross-validation. The log likelihood ratio (LLR) is used to compare each pair of models: LLR(model 1, model 2) = log(Pr(observed firing | model 1)/Pr(observed firing | model 2)). The positive values in Table 1 indicate that the non-parametric models do a better job of explaining new data than the parametric models with the Poisson+Robust fit providing the best description of the data. This P+R model implies that the conditional firing rate is well described by regions of smooth activity with relatively sharp discontinuities between them. The non-parametric models reduce the strong bias of the parametric models with a slight increase in variance and hence achieve a lower total error.

# 4 Temporal Inference

Given neural measurements our goal is to infer the motion of the hand over time. Related approaches have exploited simple linear filtering methods which do not provide a probabilistic interpretation of the data that can facilitate analysis and support the principled combination of multiple sources of information. Related probabilistic approaches have exploited Kalman filtering [2]. We note here however, that the learned models of neural activity are not-Gaussian and the dynamics of the hand motion may be non-linear. Furthermore with a small number of cells, our interpretation of the neural data may be ambiguous and the posterior probability of the kinematic variables, given the neural activity, may be best modeled by a non-Gaussian, multi-modal, distribution. To cope with these issues in a sound probabilistic framework we exploit a non-parametric approach that uses factored sampling to discretely approximate the posterior distribution, and particle filtering to propagate and update this distribution over time [8].

Let the state of the system be $\mathbf{s}_t = [r, \theta]$ at time $t$. Let $c_t^{(i)}$ be the mean firing rate of cell $i$ at time $t$ where the mean firing rate is estimated within *non-overlapping* 50 ms temporal windows. Also, let $\mathbf{c}_t = [c_t^{(1)} \ldots c_t^{(n)}]$ represent the firing rate of all $n$ cells at time $t$. Similarly let $C_t^{(i)}$ represent the sequence of these firing rates for cell $i$ up to time $t$ and let $\mathbf{C}_t = [C_t^{(1)} \ldots C_t^{(n)}]$ represent the firing of all $n$ cells up to time $t$.

We assume that the temporal dynamics of the states, $\mathbf{s}_t$, form a Markov chain for which the state at time $t$ depends only on the state at the previous time instant:

$$p(\mathbf{s}_t \,|\, \mathbf{S}_{t-1}) \;=\; p(\mathbf{s}_t \,|\, \mathbf{s}_{t-1}) \;,$$

where $\mathbf{S}_t = [\mathbf{s}_t, ..., \mathbf{s}_0]$ denotes the state history. We also assume that given $\mathbf{s}_t$, the current observation $\mathbf{c}_t$ and the previous observations $\mathbf{C}_{t-1}$ are independent.

Using Bayes rule and the above assumptions, the probability of observing the state at time $t$ given the history of firing can be written as

$$p(\mathbf{s}_t|\mathbf{C}_t) = \kappa_2 \, p(\mathbf{c}_t|\mathbf{s}_t) \, p(\mathbf{s}_t|\mathbf{C}_{t-1}) \qquad (2)$$

where $\kappa_2$ is a normalizing term that insures that the distribution integrates to one. The likelihood term $p(\mathbf{c}_t|\mathbf{s}_t) = \prod_{i=1}^{n} p(c_t^{(i)}|\mathbf{s}_t)$ assumes conditional independence of the individual cells where the likelihood for the firing rate of an individual cell is taken to be a Poisson distribution with the mean firing rate for the speed and velocity given by $\mathbf{s}_t$ determined by the conditional firing models learned in the previous section. Plotting this likelihood term for a range of states reveals that its structure is highly non-Gaussian with multiple peaks.

The temporal prior term, $p(\mathbf{s}_t|\mathbf{C}_{t-1})$ can be written as

$$p(\mathbf{s}_t|\mathbf{C}_{t-1}) \;=\; \int p(\mathbf{s}_t|\mathbf{s}_{t-1}) \, p(\mathbf{s}_{t-1}|\mathbf{C}_{t-1}) \, d\,\mathbf{s}_{t-1} \;, \qquad (3)$$

where $p(\mathbf{s}_t|\mathbf{s}_{t-1})$ embodies the temporal dynamics of the hand velocity which are assumed to be constant with Gaussian noise; that is, a diffusion process. Note, $p(\mathbf{s}_{t-1}|\mathbf{C}_{t-1})$ is the posterior distribution over the state space at time $t-1$.

The posterior, $p(\mathbf{s}_t|\mathbf{C}_t)$, is represented with a discrete, weighted set, of 3000 random samples which are propagated in time using a standard particle filter (see [8] for details). Unlike previous applications of particle filtering, the likelihood of firing for an individual cell in

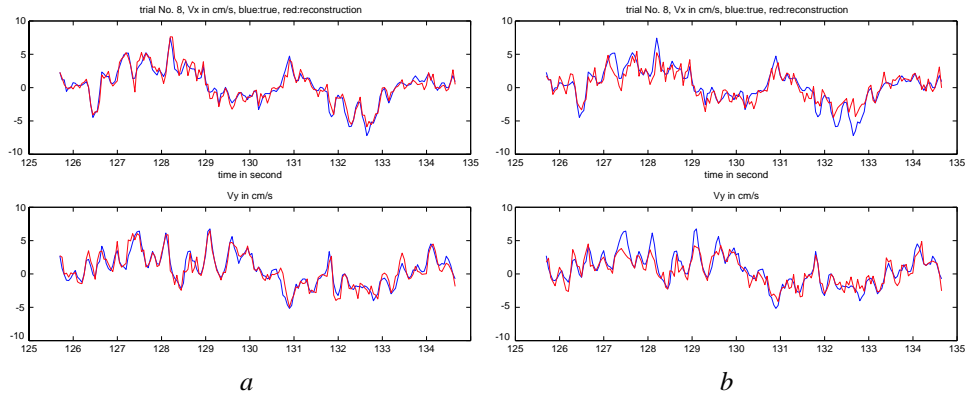

Figure 6: Tracking results using 1008 synthetic cells showing horizontal velocity, $v_x$, (top) and vertical velocity, $v_y$, (bottom). Blue indicates true velocity of hand. *(a)* Bayesian estimate using particle filtering. Red curve shows expected value of the posterior. The error is $r^2 = 0.8934$ for $v_x$ and $r^2 = 0.9033$ for $v_y$. *(b)* Linear filtering method shown in red; $r^2 = 0.7735$ for $v_x$ and $r^2 = 0.8096$ for $v_y$.

50 ms provides very little information. For the posterior to be meaningful we must combine evidence from multiple cells. Our experiments indicate that the responses from our 24 cells are insufficient for this task. To demonstrate the feasibility of the particle filtering method, we synthesized approximately 1000 cells by taking the learned models of the 24 cells and translating them along the $\theta$ axis to generate a more complete covering of the velocity space. Note that the assumption of such a set of cells in MI is quite reasonable give the sampling of cells we have observed in multiple monkeys.

From the set of synthetic cells we then generate a synthetic spike train by taking a known sequence of hand velocities and stochastically generating spikes using the learned conditional firing models with a Poisson generative model. Particle filtering is used to estimate the posterior distribution over hand velocities given the synthetic neural data. The expected value of the horizontal and vertical velocity is displayed in Figure 6a. For comparison, a standard linear filtering method [6, 14] was trained on the synthetic data from 50 ms intervals. The resulting prediction is shown in Figure 6b. Linear filtering works well over longer time windows which introduce lag. The Bayesian analysis provides a probabilistic framework for sound causal estimates over short time intervals.

We are currently experimenting with modified particle filtering schemes in which linear filtering methods provide a proposal distribution and importance sampling is used to construct a valid posterior distribution. We are also comparing these results with those of various Kalman filters.

## 5 Conclusions

We have described a Bayesian model for neural activity in MI that relates this activity to actions in the world. Quantitative comparison with previous models of MI activity indicate that the non-parametric models computed using regularization more accurately describe the neural activity. In particular, the robust spatial prior term suggests that neural firing in MI is not a smooth function of velocity but rather exhibits discontinuities between regions

of high and low activity.

We have also described the Bayesian decoding of hand motion from firing activity using a particle filter. Initial results suggest that measurements from several hundred cells may be required for accurate estimates of hand velocity. The application of particle filtering to this problem has many advantages as it allows complex, non-Gaussian, likelihood models that may incorporate non-linear temporal properties of neural firing (e.g. refractory period). Unlike previous linear filtering methods this Bayesian approach provides probabilistically sound, causal, estimates in short time windows of 50ms. Current work is exploring correlations between cells [7] and the relationship between the neural activity and other kinematic variables [12].

**Acknowledgments.** This work was supported by the Keck Foundation and by the National Institutes of Health under grants #R01 NS25074 and #N01-NS-9-2322 and by the National Science Foundation ITR Program award #0113679. We are very grateful to M. Serruya, M. Fellows, L. Paninski, and N. Hatsopoulos who provided the neural data and valuable insight.

## Footnotes

[1]By 'Gaussian+Gaussian' we mean both the likelihood and prior terms are Gaussian whereas 'Poisson+Robust' implies a Poisson likelihood and robust spatial prior.

# References

[1] M. Black and A. Rangarajan. On the unification of line processes, outlier rejection, and robust statistics with applications in early vision. *IJCV*, 19(1):57–92, 1996.

[2] E. Brown, L. Frank, D. Tang, M. Quirk, and M. Wilson. A statistical paradigm for neural spike train decoding applied to position prediction from ensemble firing patterns of rat hippocampal place cells. *J. Neuroscience*, 18(18):7411–7425, 1998.

[3] Q-G. Fu, D. Flament, J. Coltz, and T. Ebner. Temporal encoding of movement kinematics in the discharge of primate primary motor and premotor neurons. *J. of Neurophysiology*, 73(2):836–854, 1995.

[4] S. Geman and D. Geman. Stochastic relaxation, Gibbs distributions and Bayesian restoration of images. *PAMI*, 6(6):721–741, November 1984.

[5] S. Geman and D. McClure. Statistical methods for tomographic image reconstruction. *Bulletin of the Int. Stat. Inst.*, LII-4:5–21, 1987.

[6] A. Georgopoulos, A. Schwartz, and R. Kettner. Neuronal population coding of movement direction. *Science*, 233:1416–1419, 1986.

[7] N. Hatsopoulos, C. Ojakangas, L. Paninski, and J. Donoghue. Information about movement direction obtained from synchronous activity of motor cortical neurons. *Proc. Nat. Academy of Sciences*, 95:15706–15711, 1998.

[8] M. Isard and A. Blake. Condensation – conditional density propagation for visual tracking. *IJCV*, 29(1): 5–28, 1998.

[9] E. Maynard, N. Hatsopoulos, C. Ojakangas, B. Acuna, J. Sanes, R. Normann, and J. Donoghue. Neuronal interaction improve cortical population coding of movement direction. *J. of Neuroscience*, 19(18):8083–8093, 1999.

[10] D. Moran and A. Schwartz. Motor cortical representation of speed and direction during reaching. *J. Neurophysiol*, 82:2676-2692, 1999.

[11] R. Nowak and E. Kolaczyk. A statistical multiscale framework for Poisson inverse problems. *IEEE Inf. Theory*, 46(5):1811–1825, 2000.

[12] L. Paninski, M. Fellows, N. Hatsopoulos, and J. Donoghue. Temporal tuning properties for hand position and velocity in motor cortical neurons. *submitted, J. Neurophysiology*, 2001.

[13] D. Terzopoulos. Regularization of inverse visual problems involving discontinuities. *PAMI*, 8(4):413–424, 1986.

[14] J. Wessberg, C. Stambaugh, J. Kralik, P. Beck, M. Laubach, J. Chapin, J. Kim, S. Biggs, M. Srinivasan, and M. Nicolelis. Real-time prediction of hand trajectory by ensembles of cortical neurons in primates. *Nature*, 408:361–365, 2000.
